# An iterative improvement procedure for hierarchical clustering

**David Kauchak**
Department of Computer Science
University of California, San Diego
dkauchak@cs.ucsd.edu

**Sanjoy Dasgupta**
Department of Computer Science
University of California, San Diego
dasgupta@cs.ucsd.edu

## Abstract

We describe a procedure which finds a hierarchical clustering by hill-climbing. The cost function we use is a hierarchical extension of the $k$-means cost; our local moves are tree restructurings and node reorderings. We show these can be accomplished efficiently, by exploiting special properties of squared Euclidean distances and by using techniques from scheduling algorithms.

## 1  Introduction

A *hierarchical clustering* of $n$ data points is a recursive partitioning of the data into $2, 3, 4, \ldots$ and finally $n$ clusters. Each intermediate clustering is made more fine-grained by splitting one of its clusters. It is natural to depict this process as a tree whose leaves are the data points and whose interior nodes represent intermediate clusters. Such hierarchical representations are very popular – they depict a data set at multiple levels of granularity, simultaneously; they require no prior specification of the number of the clusters; and there are several simple heuristics for constructing them [2, 3].

Some of these heuristics – such as *average-linkage* – implicitly try to create clusters of small "radius" throughout the hierarchy. However, to the best of our knowledge, there is so far no procedure which specifically hillclimbs the space of hierarchical clusterings according to a precise objective function. Given the heuristic nature of existing algorithms, it would be most helpful to be able to call an iterative improvement procedure on their output. In particular, we seek an analogue of $k$-means for hierarchical clustering. Taken literally this is possible only to a certain extent – the basic object we are dealing with is a tree rather than a partition – but $k$-means has closely informed many aspects of our procedure, and has determined our choice of objective function.

We use a canonical tree representation of a hierarchical clustering, in which the leaves are data points, and the interior nodes are ordered; such a clustering is specified completely by a tree structure and by an ordering of nodes. Our cost function is a hierarchical extension of the $k$-means cost function, and is the same cost function which motivates average-linkage schemes. Our iterative procedure alternates between two simple moves:

1. The ordering of nodes is kept fixed, and one subtree is relocated. This is the natural generalization of a standard heuristic clustering move in which a data point is transferred from one cluster to another.

2. The tree structure is kept fixed, and its interior nodes are reordered optimally.

We show that by exploiting properties of Euclidean distance (which underlies the $k$-means cost function and therefore ours as well), these tasks can be performed efficiently. For instance, the second one can be transformed into a problem in VLSI design and job scheduling called *minimum linear arrangement*. In general this problem is NP-hard, but for our particular case it is known [4] to be efficiently solvable, in $O(n \log n)$ time. After motivating and describing our model and our algorithm, we end with some experimental results.

## 2 The model

### 2.1 The space of trees

A hierarchical clustering of $n$ points contains $n$ different clusterings, nested within each other. It is often depicted using a *dendogram*, such as the one below on the left (for a data set of five points). We will use the term *k-clustering*, and the notation $\mathbb{C}_k$, to denote the grouping into $k$ clusters. One of these clusters is divided in two to yield the $(k+1)$-clustering $\mathbb{C}_{k+1}$, and so on. Instead of a dendogram, it is convenient to use a rooted binary tree (shown below on the right) in which the leaves are data points and internal nodes have exactly two children, so there are $2n - 1$ nodes overall. Each internal node is annotated with a unique "split number" between 1 and $n - 1$. These satisfy the property that the split number of a parent is less than that of its children; so the root is numbered 1. The $k$-clustering is produced by removing the internal nodes numbered $1, 2, 3, \ldots, k - 1$; each cluster consists of (the leaves in) one of the resulting connected components.

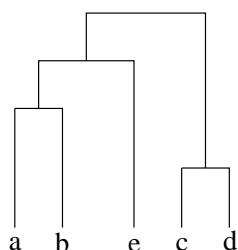

- 2-clustering:
  $\{a, b, e\}, \{c, d\}$

- 3-clustering:
  $\{a, b\}, \{e\}, \{c, d\}$

- 4-clustering:
  $\{a\}, \{b\}, \{e\}, \{c, d\}$

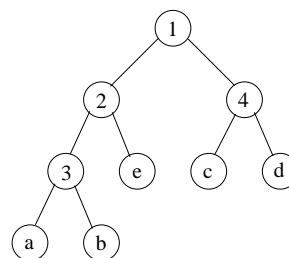

Henceforth we will use "node $i$" to mean "the internal node with split number $i$". The maximal subtree rooted at this node is $T_i$; the mean of its data points (leaves) is called $\mu_i$.

To summarize, a hierarchical clustering is specified by: a binary tree with the data points at the leaves; and an ordering of the internal nodes.

### 2.2 Cost function

If the clusters of $\mathbb{C}_k$ are $S_1, S_2, \ldots, S_k$, then the $k$-means cost function is

$$\text{cost}(\mathbb{C}_k) \ = \ \sum_{j=1}^{k} \sum_{x \in S_j} \|x - \mu(S_j)\|^2,$$

where $\mu(S)$ is the mean of set $S$. To evaluate a hierarchical clustering, we need to combine the costs of all $n$ intermediate clusterings, and we do so in the most obvious way, by a linear combination. We take the overall cost of the hierarchical clustering to be

$$\sum_{k=1}^{n} w_k \cdot \text{cost}(\mathbb{C}_k),$$

where the $w_k$ are non-negative weights which add up to one. The default choice is to make all $w_k = 1/n$, but in general the specific application will dictate the choice of weights. A decreasing schedule $w_1 > w_2 > w_3 > \cdots > w_n$ places more emphasis upon coarser clusterings (ie. small $k$); a setting $w_k = 1$ singles out a particular intermediate clustering.

Although many features of our cost function are familiar from the simpler $k$-means setting, there is one which is worth pointing out. Consider the set of six points shown here:

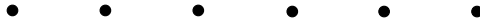

Under the $k$-means cost function, it is clear what the best 2-clustering is (three points in each cluster). It is similarly clear what the best 3-clustering is, but this cannot be nested within the best 2-clustering. In other words, the imposition of a hierarchical structure forces certain tradeoffs between the intermediate clusterings. This particular feature is fundamental to hierarchical clustering, and in our cost function it is laid bare. By adjusting the weights $w_k$, the user can bias this tradeoff according to his or her particular needs.

It is worth pointing out that $\mathrm{cost}(\mathbb{C}_k)$ decreases as $k$ increases; as more clusters are allowed, the data can be modeled with less error. This means that even when all the weights $w_k$ are identical, the smaller values of $k$ contribute more to the cost function, and therefore, a procedure for minimizing this function must implicitly focus a little more on smaller $k$ than on larger $k$. This is the sort of bias we usually seek. If we wanted to further emphasize small values of $k$, we could for instance use an exponentially decreasing schedule of weights, ie. $w_k = c \cdot \alpha^k$, where $\alpha < 1$ and where $c$ is a normalization constant.

Notice that any given subtree $T_j$ can appear as an individual cluster in many of the clusterings $\mathbb{C}_k$. If $\pi(j)$ denotes the parent of $j$, then $T_j$ first appears as its own cluster in $\mathbb{C}_{\pi(j)+1}$, and is part of all the successive clusterings up to and including $\mathbb{C}_j$. At that point, it gets split in two.

## 2.3 Relation to previous work

The most commonly used heuristics for hierarchical clustering are *agglomerative*. They work bottom-up, starting with each data point in its own cluster, and then repeatedly merging the two "closest" clusters until finally all the points are grouped together in one cluster. The different schemes are distinguished by their measure of closeness between clusters.

1. *Single linkage* – the distance between two clusters $S$ and $T$ is taken to be the distance between their closest pair of points, ie. $\min_{x \in S, y \in T} \|x - y\|$.

2. *Complete linkage* uses the distance between the farthest pair of points, ie. $\max_{x \in S, y \in T} \|x - y\|$.

3. *Average linkage* seems to have now become a generic term encompassing at least three different measures of distance between clusters.

   (a) (Sokal-Michener) $\|\mu(S) - \mu(T)\|^2$
   (b) $\frac{1}{|S| \cdot |T|} \sum_{x \in S, y \in T} \|x - y\|^2$
   (c) (Ward's method) $\frac{|S| \cdot |T|}{|S| + |T|} \|\mu(S) - \mu(T)\|^2$

Average linkage appears to be the most widely used of these; for instance, it is a standard tool for analyzing gene expression data [1]. The three average linkage distance functions are all trying to minimize something very much like our cost function. In particular, Ward's measure of the distance between two clusters is exactly the increase in $k$-means cost occasioned by merging those clusters. For our experimental comparisons, we have therefore chosen Ward's method.

## 3 Local moves

Each element of the search space is a tree structure in which the data points are leaves and in which the interior nodes are ordered. A quick calculation shows that this space has size $n((n-1)!)^2/2^{n-1}$ (consider the sequence of $n-1$ merge operations which create the tree from the data set). We consider two moves for navigating the space, along the lines of the standard "alternating optimization" paradigm of $k$-means and EM:

1. keep the structure fixed and reorder the internal nodes optimally;
2. keep the ordering of the internal nodes fixed and alter the structure by relocating some subtree.

A key concern in the design of these local moves is *efficiency*. A $k$-means update takes $O(kn)$ time; in our situation the analogue would be $O(n^2)$ time since we are dealing with all values of $k$. Ideally, however, we'd like a faster update. For our first move – reordering internal nodes – we show that a previously-known scheduling algorithm [4] can be adapted to solve this task (in the case of uniform weights) in just $O(n \log n)$ time. For the second move, we show that any given subtree can be relocated optimally in $O(n)$ time, using just a single pass through the tree. These efficiency results are nontrivial; a crucial step in obtaining them is to exploit special properties of squared Euclidean distance. In particular, we write our cost function in three different, but completely equivalent, ways; and we switch back and forth between these:

1. In the form given above (the definition).
2. We define the cost of a subtree $T_i$ to be $\text{cost}(T_i) = \sum_{x \in T_i} \|x - \mu_i\|^2$ (where the sum is over leaf nodes), that is, the cost of the single cluster rooted at point $i$. Then the overall cost is a linear combination of subtree costs. Specifically, it is

$$\sum_{j=1}^{n-1} W_{\pi(j),j} \cdot \text{cost}(T_j),  \tag{1}$$

   where $\pi(j)$ is the parent of node $j$ and $W_{ij} = w_{i+1} + w_{i+2} + \cdots + w_j$.
3. We annotate each tree edge $(i,j)$ ($i$ is the parent of $j > i$) by $\|\mu_i - \mu_j\|^2$; the overall cost is also a linear combination of these edge weights, specifically,

$$\sum_{(k,l) \in T} W_k \cdot n_l \cdot \|\mu_k - \mu_l\|^2,  \tag{2}$$

   where $W_k = w_1 + w_2 + \cdots + w_k$ and $n_l$ is the number of leaves in subtree $T_l$.

All proofs are in a technical report [5] which can be obtained from the authors. To give a hint for why these alternative formulations of the cost function are true, we briefly mention a simple "bias-variance" decomposition of squared Euclidean distance:

Suppose $S$ is a set of points with mean $\mu_S$. Then for any $\mu$,

$$\sum_{x \in S} \|x - \mu\|^2 = \sum_{x \in S} \|x - \mu_S\|^2 + |S| \cdot \|\mu - \mu_S\|^2.$$

### 3.1 The graft

In a *graft* move, an entire subtree is moved to a different location, as shown below. The letters $a, b, i, \ldots$ denote split numbers of interior nodes; here the subtree $T_j$ is moved. The only prerequisite (to ensure a consistent ordering) is $a < i < b$.

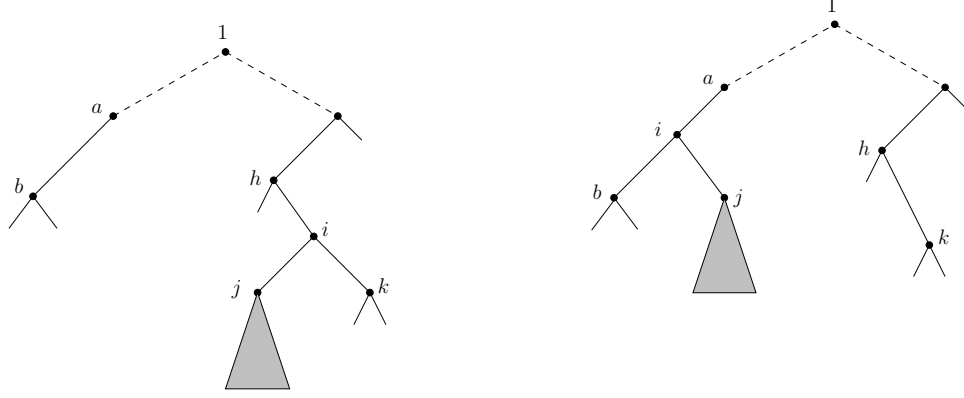

First of all, a basic sanity check: this move enables us to traverse the entire search space.

**Claim.** Any two hierarchical clusterings are connected by a sequence of *graft* operations.

It is important to find good grafts *efficiently*. Suppose we want to move a subtree $T_j$; what is the best place for it? Evaluating the cost of a hierarchical clustering takes $O(n)$ time using equation (1) and doing a single, bottom-up pass. Since there are $O(n)$ possible locations for $T_j$, naively it seems like evaluating all of them would take $O(n^2)$ time. In fact, the best relocation of $T_j$ can be computed in just $O(n)$ time, in a single pass over the tree.

To see why this is possible, notice that in the diagram above, the movement of $T_j$ affects only the subtrees on the path between $a$ and $h$. Some of these subtrees get bigger ($T_j$ is added to them); others shrink ($T_j$ is removed). The precise change in cost of any given subtree $T_l$ on this path is easy to compute:

**Claim.** If subtree $T_j$ is merged into $T_l$, then the cost of $T_l$ goes up by

$$\Delta_l^+ = \text{cost}(T_l \cup T_j) - \text{cost}(T_l) = \text{cost}(T_j) + \frac{n_l n_j}{n_l + n_j} \cdot \|\mu_l - \mu_j\|^2.$$

**Claim.** If subtree $T_j \subset T_l$ is removed from $T_l$, then the cost of $T_l$ changes by

$$\Delta_l^- = \text{cost}(T_l - T_j) - \text{cost}(T_l) = -\text{cost}(T_j) - \frac{n_i n_l}{n_l - n_j} \cdot \|\mu_l - \mu_j\|^2.$$

Using (1), the total change in cost from grafting $T_j$ between $a, b$ (as depicted above) can be found by adding terms of the form $W_{\pi(l),l}\Delta_l^{\pm}$, for nodes $l$ on the path between $j$ and $a$. This suggests a two-pass algorithm for optimally relocating $T_j$: in the first pass over the tree, for each $T_l$, the potential cost change from adding/removing $T_j$ is computed. The second pass finds the best location. In fact, these can be combined into a single pass [5].

### 3.2 Reordering internal nodes

Let $V_{int}$ be the interior nodes of the tree; if there are $n$ data points (leaves), then $|V_{int}| = n - 1$. For any $x \in V_{int}$, let $T_x$ be the maximal subtree rooted at $x$, which contains all the descendants of $x$. Let $n_x$ be the number of leaves in this subtree. If $x$ has children $y$ and $z$, then the *goodness of split* at $x$ is the reduction in cost obtained by splitting cluster $T_x$,

$$\text{cost}(T_x) - (\text{cost}(T_y) + \text{cost}(T_z)),$$

which we henceforth denote $g(x)$ (for leaves $g(x) = 0$). Again using properties of Euclidean distance, we can rewrite it thus:

$$g(x) = n_y \|\mu_x - \mu_y\|^2 + n_z \|\mu_x - \mu_z\|^2.$$

Priority queue operations:
makequeue, max, deletemax,
union, insert.

Linked list operations:
∘ (concatenation)

procedure reorder(T)
$u \leftarrow$ root of $T$
$Q \leftarrow$ makequeue($u$)
while $Q$ is not empty
    $L \leftarrow$ deletemax($Q$)
    Output elements of list $L$, in order

function makequeue($x$)
if $x$ is a leaf return { }
let $y, z$ be the children of $x$
$Q \leftarrow$ union(makequeue($y$), makequeue($z$))
$r \leftarrow n_y \|\mu_x - \mu_y\|^2 + n_z \|\mu_x - \mu_z\|^2$
$L \leftarrow [x]$

while $r < r(\max(Q))$
    $L' \leftarrow$ deletemax($Q$)
    $r \leftarrow \frac{r \cdot |L| + r(L') \cdot |L'|}{|L| + |L'|}$
    $L \leftarrow L \circ L'$
$r(L) \leftarrow r$
insert($Q, L$)
return $Q$

Figure 1: The reordering move. Here $Q$ is a priority queue of linked lists. Each list $L$ has a value $r(L)$; and $Q$ is ordered according to these.

We wish to find a numbering $\sigma : V_{int} \rightarrow \{1, 2, \ldots, n-1\}$ which

– respects the precedence constraints of the tree: if $x$ is the parent of $y$ then $\sigma(x) < \sigma(y)$.

– minimizes the overall cost of the hierarchical clustering. Assuming uniform weights $w_k = 1/n$, this cost can be seen (by manipulating equation (2)) to be

$$\frac{1}{n} \sum_{x \in V_{int}} \sigma(x) g(x).$$

Notice that this is essentially a scheduling problem. There is a "task" (a split) corresponding to each $x \in V_{int}$. We would like to schedule the good tasks (with high $g(x)$) early on; in the language of clustering, if there are particularly useful splits (which lead to well separated clusters), we would like to perform them early in the hierarchy. And there are precedence constraints which must be respected: certain splits must precede others.

The naive greedy solution – always pick the node with highest $g(x)$, subject to precedence constraints – doesn't work. The reason: it is quite possible that a particular split has low $g(x)$-value, but that it leads to other splits of very high value. A greedy algorithm would schedule this split very late; an algorithm with some "lookahead" capability would realize the value of this split and schedule it early.

Horn[4] has a scheduling algorithm which obtains the optimal ordering, in the case where all the weights $w_k$ are equal, and can be implemented in $O(n \log n)$ time. We believe it can be extended to exponentially decaying, "memoryless" weights, ie. $w_k = c \cdot \alpha^k$, where $\alpha < 1$ and $c$ is some normalization constant.

We now present an overview of Horn's algorithm. For each node $x \in V$, define $r(x)$ to be the maximum, over all subtrees $T$ (not necessarily maximal) rooted at $x$, of $\frac{1}{|T|} \sum_{z \in T} g(z)$ (in words, the average of $g(\cdot)$ over nodes of $T$). This value $r(x)$ is a more reliable indicator of the utility of split $x$ than the immediate return $g(x)$. Once these $r(x)$ are known, the optimal numbering is easy to find: pick nodes in decreasing order of $r(\cdot)$ while respecting the precedence constraints. So the main goal is to compute the $r(x)$ for all $x$ in the tree. This can be done by a short divide-and-conquer procedure in $O(n \log n)$ time (Figure 1).

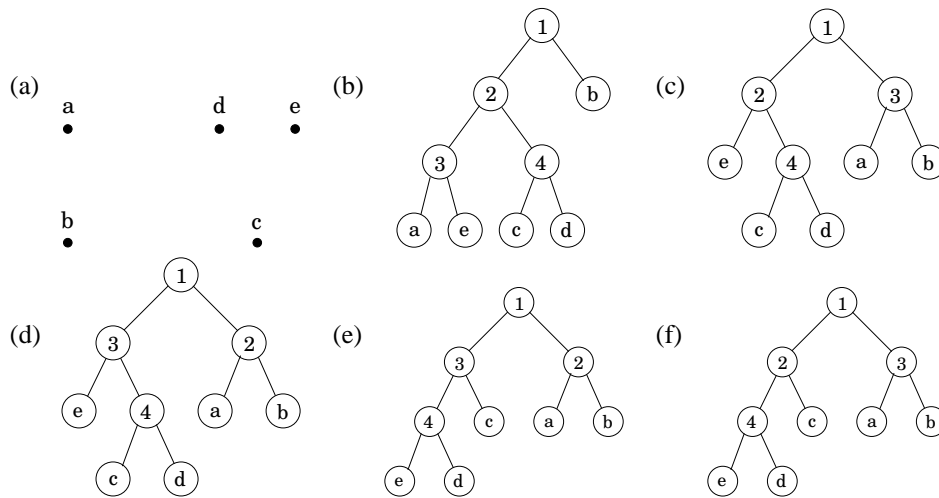

Figure 2: (a) Five data points. (b)–(f) Iteratively improving the hierarchical clustering.

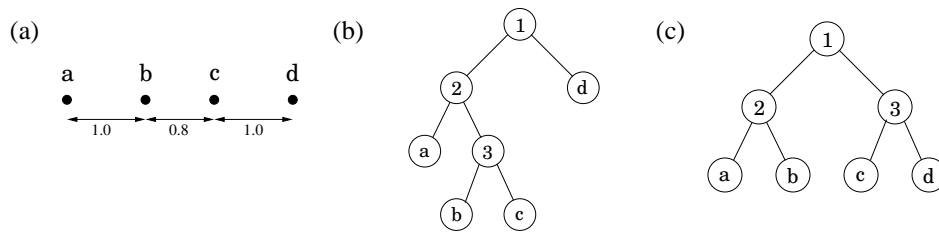

Figure 3: (a) Four points on a line. (b) Average linkage. (c) Optimal tree.

## 4 Experiments

In the experiments, we used uniform weights $w_k = 1/n$. In each iteration of our procedure, we did a reordering of the nodes, and performed one graft – by trying each possible subtree (all $O(n)$ of them), determining the optimal move for that subtree, and greedily picking the best move. We would prefer a more efficient, randomized way to pick which subtree to graft – either completely randomly, or biased by a simple criterion like "amount it deviates from the center of its parent cluster"; this is future work.

**Simple examples.** To give some concrete intuition, Figure 2 shows the sequence of moves taken on a toy example involving five data points in the plane. The initial tree (b) is random and has a cost of 62.25. A single graft (c) reduces the cost to 27. A reordering (d), swapping 2 and 3, reduces the cost to 25.5, and a further graft (e) and reordering (f) result in the final tree, which is optimal and has cost 21.

Figure 3 demonstrates a typical failing of average linkage. The initial greedy merger of points $b, c$ gives a small early benefit but later turns out to be a bad idea; yet the resulting tree is only one graft away from being optimal. Really bad cases for average linkage can be constructed by recursively compounding this simple instance.

**A larger data set.** Average linkage is often used in the analysis of gene expression data.

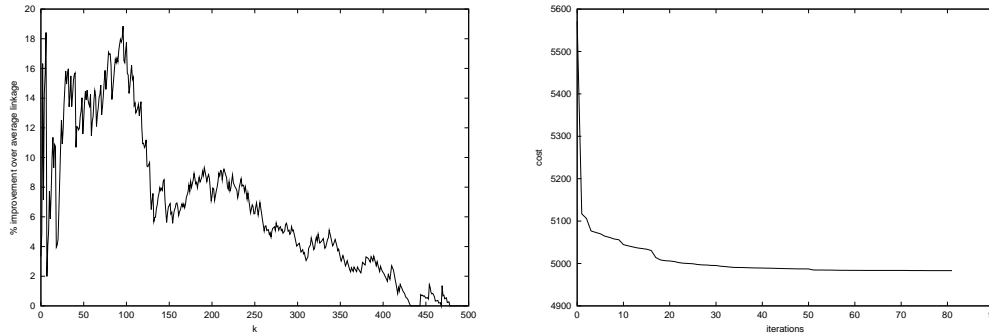

Figure 4: (a) On the left, a comparison with average linkage. (b) On the right, the behavior of the cost function over the 80 iterations required for convergence.

We tried our method on the yeast data of [1]. We randomly chose clean subsets (no missing entries) of varying sizes from this data set, and tried the following on it: average linkage, our method initialized randomly, and our method initialized with average linkage.

There were two clear trends. First of all: our method, whether initialized randomly or with average linkage, systematically did better than average linkage, not only for the particular aggregate cost function we are using, but across the whole spectrum of values of $k$. Figure 4(a), obtained on a 500-point data set, shows for each $k$, the percent by which the (induced) $k$-clustering found in our method (initialized with average linkage) improved upon that found by average linkage; the metric here is the $k$-means cost function. This is a fair comparison because both methods are explicitly trying to minimize this cost. Notice that an improvement in the aggregate (weighted average) is to be expected, since we are hillclimbing based on this measure. What was reassuring to us was that this improvement came across at almost all values of $k$ (especially the smaller ones), rather than by negotiating some unexpected tradeoff between different values of $k$. This experiment also indicates that, in general, the output of average linkage has real scope for improvement.

Second, our method often took an order of magnitude (ten or more times) longer to converge if initialized randomly than if initialized with average linkage, even though better solutions were often found with random initialization. We therefore prefer starting with average linkage. On the scant examples we tried, there was a period of rapid improvement involving grafts of large subtrees, followed by a long series of minor "fixes"; see Figure 4(b), which refers again to the 500-point data set mentioned earlier.

## References

[1] T.L. Ferea *et al*. Systematic changes in gene expression patterns following adaptive evolution in yeast. *Proceedings of the National Academy of Sciences*, 97, 1999.

[2] J.A. Hartigan. *Clustering algorithms*. Wiley, 1975.

[3] J.A. Hartigan. Statistical theory in clustering. *Journal of Classification*, 1985.

[4] W.A. Horn. Single-machine job sequencing with treelike precedence ordering and linear delay penalties. *SIAM Journal on Applied Mathematics*, 23:189–202, 1972.

[5] D. Kauchak and S. Dasgupta. *Manuscript*, 2003.
